# Monte Carlo Matrix Inversion and Reinforcement Learning

Andrew Barto and Michael Duff
Computer Science Department
University of Massachusetts
Amherst, MA 01003

## Abstract

We describe the relationship between certain reinforcement learning (RL) methods based on dynamic programming (DP) and a class of unorthodox Monte Carlo methods for solving systems of linear equations proposed in the 1950's. These methods recast the solution of the linear system as the expected value of a statistic suitably defined over sample paths of a Markov chain. The significance of our observations lies in arguments (Curtiss, 1954) that these Monte Carlo methods scale better with respect to state-space size than do standard, iterative techniques for solving systems of linear equations. This analysis also establishes convergence rate estimates. Because methods used in RL systems for approximating the evaluation function of a fixed control policy also approximate solutions to systems of linear equations, the connection to these Monte Carlo methods establishes that algorithms very similar to TD algorithms (Sutton, 1988) are asymptotically more efficient in a precise sense than other methods for evaluating policies. Further, all DP-based RL methods have some of the properties of these Monte Carlo algorithms, which suggests that although RL is often perceived to be slow, for sufficiently large problems, it may in fact be more efficient than other known classes of methods capable of producing the same results.

# 1  Introduction

Consider a system whose dynamics are described by a finite state Markov chain with transition matrix $P$, and suppose that at each time step, in addition to making a transition from state $x_t = i$ to $x_{t+1} = j$ with probability $p_{ij}$, the system produces a randomly determined reward, $r_{t+1}$, whose expected value is $R_i$. The *evaluation function*, $V$, maps states to their expected, infinite-horizon discounted returns:

$$V_i = E\left\{\sum_{t=0}^{\infty}\gamma^t r_{t+1}|x_0 = i\right\}.$$

It is well known that $V$ uniquely satifies a linear system of equations describing local consistency:

$$V = R + \gamma PV,$$

or

$$(I - \gamma P)V = R. \tag{1}$$

The problem of computing or estimating $V$ is interesting and important in its own right, but perhaps more significantly, it arises as a (rather computationally-burdensome) step in certain techniques for solving Markov Decision Problems. In each iteration of Policy-Iteration (Howard, 1960), for example, one must determine the evaluation function associated with some fixed control policy, a policy that improves with each iteration.

Methods for solving (1) include standard iterative techniques and their variants—successive approximation (Jacobi or Gauss-Seidel versions), successive over-relaxation, etc. They also include some of the algorithms used in reinforcement learning (RL) systems, such as the family of TD algorithms (Sutton, 1988). Here we describe the relationship between the latter methods and a class of unorthodox Monte Carlo methods for solving systems of linear equations proposed in the 1950's. These methods recast the solution of the linear system as the expected value of a statistic suitably defined over sample paths of a Markov chain.

The significance of our observations lies in arguments (Curtiss, 1954) that these Monte Carlo methods scale better with respect to state-space size than do standard, iterative techniques for solving systems of linear equations. This analysis also establishes convergence rate estimates. Applying this analysis to particular members of the family of TD algorithms (Sutton, 1988) provides insight into the scaling properties of the TD family as a whole and the reasons that TD methods can be effective for problems with very large state sets, such as in the backgammon player of Tesauro (Tesauro, 1992).

Further, all DP-based RL methods have some of the properties of these Monte Carlo algorithms, which suggests that although RL is often slow, for large problems (Markov Decision Problems with large numbers of states) it is in fact far more practical than other known methods capable of producing the same results. First, like many RL methods, the Monte Carlo algorithms do not require explicit knowledge of the transition matrix, $P$. Second, unlike standard methods for solving systems of linear equations, the Monte Carlo algorithms can approximate the solution for *some* variables without expending the computational effort required to approximate

the solution for all of the variables. In this respect, they are similar to DP-based RL algorithms that approximate solutions to Markovian decision processes through repeated trials of simulated or actual control, thus tending to focus computation onto regions of the state space that are likely to be relevant in actual control (Barto *et. al.*, 1991).

This paper begins with a condensed summary of Monte Carlo algorithms for solving systems of linear equations. We show that for the problem of determining an evaluation function, they reduce to simple, practical implementations. Next, we recall arguments (Curtiss, 1954) regarding the scaling properties of Monte Carlo methods compared to iterative methods. Finally, we conclude with a discussion of the implications of the Monte Carlo technique for certain algorithms useful in RL systems.

## 2   Monte Carlo Methods for Solving Systems of Linear Equations

The Monte Carlo approach may be motivated by considering the statistical evaluation of a simple sum, $\sum_k a_k$. If $\{p_k\}$ denotes a set of values for a probability mass function that is arbitrary (save for the requirement that $a_k \neq 0$ imply $p_k \neq 0$), then $\sum_k a_k = \sum_k \left(\frac{a_k}{p_k}\right) p_k$, which may be interpreted as the expected value of a random variable $Z$ defined by $Pr\left\{Z = \frac{a_k}{p_k}\right\} = p_k$.

From equation (1) and the Neumann series representation of the inverse it is is clear that
$$V = (I - \gamma P)^{-1} R = R + \gamma P R + \gamma^2 P^2 R + \cdots$$
whose $i^{th}$ component is
$$V_i = R_i + \gamma \sum_{i_1} p_{ii_1} R_{i_1} + \gamma^2 \sum_{i_1 i_2} p_{ii_1} p_{i_1 i_2} R_{i_2} + \cdots$$
$$\cdots + \gamma^k \sum_{i_1 \cdots i_k} p_{ii_1} \cdots p_{i_{k-1} i_k} R_{i_k} + \cdots \tag{2}$$
and it is this series that we wish to evaluate by statistical means.

A technique originated by Ulam and von-Neumann (Forsythe & Leibler, 1950) utilizes an arbitrarily defined Markov chain with transition matrix $\tilde{P}$ and state set $\{1, 2, ..., n\}$ ($V$ is assumed to have $n$ components). The chain begins in state $i$ and is allowed to make $k$ transitions, where $k$ is drawn from a geometric distribution with parameter $p_{step}$; i.e., $Pr\{k \text{ state transitions}\} = p_{step}^k (1 - p_{step})$. The Markov chain, governed by $\tilde{P}$ and the geometrically-distributed stopping criterion, defines a mass function assigning probability to every trajectory of every length starting in state $i$, $x_0 = i_0 = i \to x_1 = i_1 \to \cdots \to x_k = i_k$, and to each such trajectory there corresponds a unique term in the sum (2).

For the case of value estimation, "$Z$" is defined by
$$Pr\left\{Z = \frac{\gamma^k \prod_{j=1}^k p_{i_{j-1} i_j} R_{i_k}}{p_{step}^k (1 - p_{step}) \prod_{j=1}^k \tilde{p}_{i_{j-1} i_j}}\right\} = p_{step}^k (1 - p_{step}) \prod_{j=1}^k \tilde{p}_{i_{j-1} i_j},$$

which for $\tilde{P} = P$ and $p_{step} = \gamma$ becomes

$$Pr\left\{Z = \frac{R_{i_k}}{1-\gamma}\right\} = \gamma^k(1-\gamma)\prod_{j=1}^{k} p_{i_{j-1}i_j}.$$

The sample average of sampled values of $Z$ is guaranteed to converge (as the number of samples grows large) to state $i$'s expected, infinite-horizon discounted return.

In Wasow's method (Wasow, 1952), the *truncated* Neumann series

$$\tilde{V}_i = R_i + \gamma\sum_{i_1}p_{ii_1}R_{i_1} + \gamma^2\sum_{i_1i_2}p_{ii_1}p_{i_1i_2}R_{i_2} + \cdots + \gamma^N\sum_{i_1\cdots i_N}p_{ii_1}\cdots p_{i_{N-1}i_N}R_{i_N}$$

is expressed as $R_i$ plus the expected value of the *sum* of $N$ random variables $Z_1, Z_2, ..., Z_N$, the intention being that

$$E(Z_k) = \gamma^k\sum_{i_1\cdots i_k}p_{ii_1}p_{i_1i_2}\cdots p_{i_{k-1}i_k}R_{i_k}.$$

Let trajectories of length $N$ be generated by the Markov chain governed by $\tilde{P}$. A given term $\gamma^k p_{ii_1}p_{i_1i_2}\cdots p_{i_{k-1}i_k}R_{i_k}$ is associated with all trajectories $i \to i_1 \to i_2 \to \cdots \to i_k \to i_{k+1} \to \cdots \to i_N$ whose first $k+1$ states are $i, i_1, ..., i_k$. The measure of this set of trajectories is just $\tilde{p}_{ii_1}\tilde{p}_{i_1i_2}\cdots\tilde{p}_{i_{k-1}i_k}$. Thus, the random variables $Z_k$, $k = 1, N$ are defined by

$$Pr\left\{Z_k = \frac{\gamma^k p_{ii_1}p_{i_1i_2}\cdots p_{i_{k-1}i_k}R_{i_k}}{\tilde{p}_{ii_1}\tilde{p}_{i_1i_2}\cdots\tilde{p}_{i_{k-1}i_k}}\right\} = \tilde{p}_{ii_1}\tilde{p}_{i_1i_2}\cdots\tilde{p}_{i_{k-1}i_k}.$$

If $\tilde{P} = P$, then the estimate becomes an average of sample truncated, discounted returns: $\tilde{V}_i = R_i + \gamma R_{i_1} + \gamma^2 R_{i_2} + \cdots + \gamma^N R_{i_N}$.

The Ulam/von Neumann approach may be reconciled with that of Wasow by processing a given trajectory *a posteriori*, converting it into a set of terminated paths consistent with any choice of stopping-state transition probabilities. For example, for a stopping state transition probability of $1-\gamma$, a path of length $k$ has probability $\gamma^k(1-\gamma)$. Each "prefix" of the observed path $x(0) \to x(1) \to x(2) \to \cdots$ can be weighted by the probability of a path of corresponding length, resulting in an estimate, $V$, that is the sampled, discounted return:

$$V = \sum_{k=0}^{\infty}\gamma^k R_{x(k)}.$$

## 3   Complexity

In (Curtiss, 1954) Curtiss establishes a theoretical comparison of the complexity (number of multiplications) required by the Ulam/von Neumann method and a stationary linear iterative process for computing a *single* component of the solution to a system of linear equations. Curtiss develops an analytic formula for bounds on the conditional mean and variance of the Monte-Carlo sample estimate, $V$, and mean and variance of a sample path's time to absorption, then appeals to the

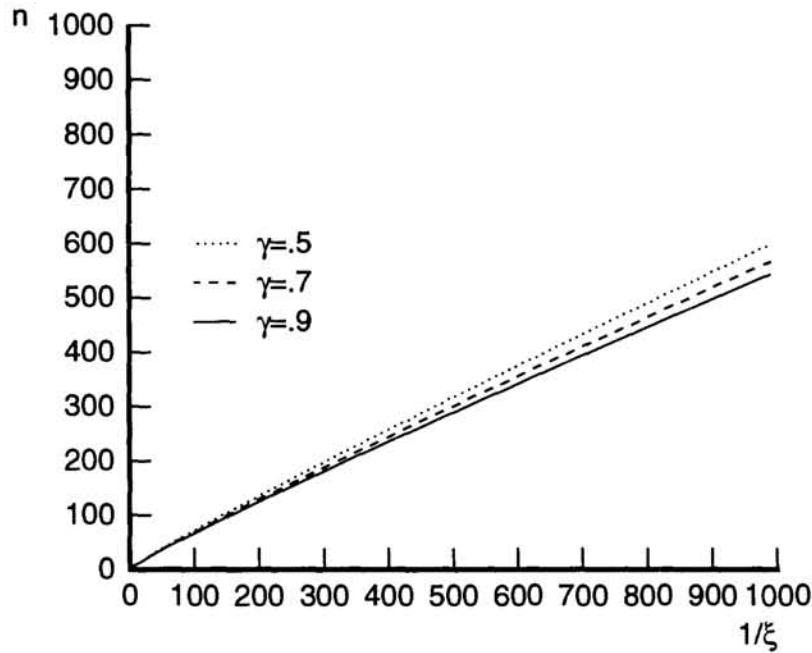

Figure 1: Break-even size of state space versus accuracy.

Central Limit Theorem to establish a 95%-confidence interval for the complexity of his method to reduce the initial error by a given factor, $\xi$. [1]

For the case of value-estimation, Curtiss' formula for the Monte-Carlo complexity may be written as

$$WORK_{Monte-Carlo} = \frac{1}{1-\gamma}\left(1 + \frac{2}{\xi^2}\right). \qquad (3)$$

This is compared to the complexity of the iterative method, which for the value-estimation problem takes the form of the classical dynamic programming recursion, $V^{(n+1)} = R + \gamma P V^{(n)}$:

$$WORK_{iterative} = \left(1 + \frac{\log \xi}{\log \gamma}\right) n^2 + n.$$

The iterative method's complexity has the form $an^2 + n$, with $a > 1$, while the Monte-Carlo complexity is *independent of n*—it is most sensitive to the amount of error reduction desired, signified by $\xi$. Thus, given a fixed amount of computation, for large enough $n$, the Monte-Carlo method is likely (with 95% confidence level) to produce better estimates. The theoretical "break-even" points are plotted in Figure 1, and Figure 2 plots work versus state-space size for example values of $\gamma$ and $\xi$.

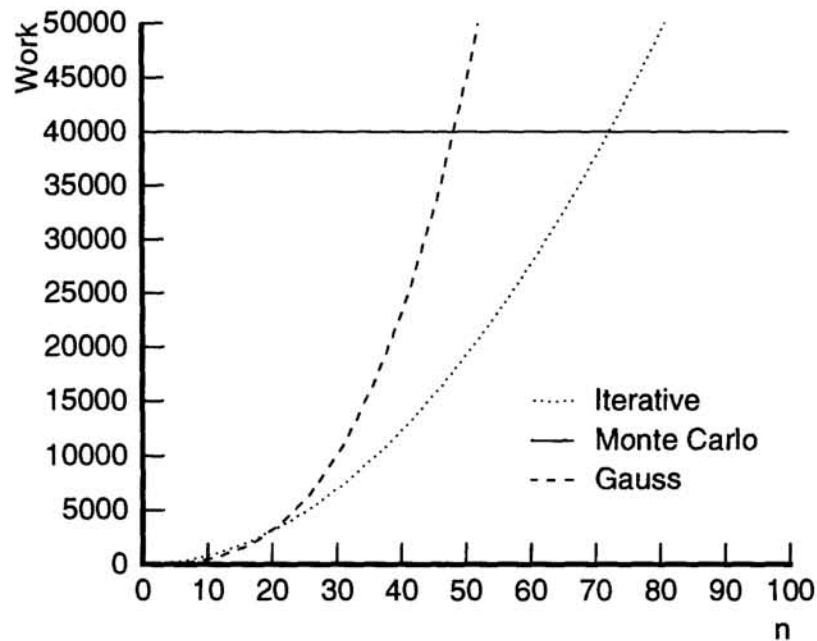

Figure 2: Work versus number of states for $\gamma = .5$ and $\xi = .01$.

## 4   Discussion

It was noted that the analytic complexity Curtiss develops is for the work required to compute *one* component of a solution vector. In the worst case, all components could be estimated by constructing $n$ separate, independent estimators. This would multiply the Monte-Carlo complexity by a factor of $n$, and its scaling supremacy would be only marginally preserved. A more efficient approach would utilize data obtained in the course of estimating one component to estimate other components as well; Rubinstein (Rubinstein, 1981) decribes one way of doing this, using the notion of "covering paths." Also, it should be mentioned that substituting more sophisticated iterative methods, such as Gauss-Seidel, in place of the simple successive approximation scheme considered here, serves only to improve the condition number of the underlying iterative operator—the amount of computation required by iterative methods remains $an^2 + n$, for some $a > 1$.

An attractive feature of the the analysis provided by Curtiss is that, in effect, it yields information regarding the convergence *rate* of the method; that is, Equation 4 can be re-arranged in terms of $\xi$. Figure 3 plots $\xi$ versus work for example values of $\gamma$ and $n$.

The simple Monte Carlo scheme considered here is practically identical to the limiting case of TD-$\lambda$ with $\lambda$ equal to one (TD-1 differs in that its averaging of sampled, discounted returns is weighted with recency). Ongoing work (Duff) explores the connection between TD-$\lambda$ (Sutton, 1988), for general values of $\lambda$, and Monte Carlo methods augmented by certain variance reduction techniques. Also, Barnard (Barnard) has noted that TD-0 may be viewed as a stochastic approxima-

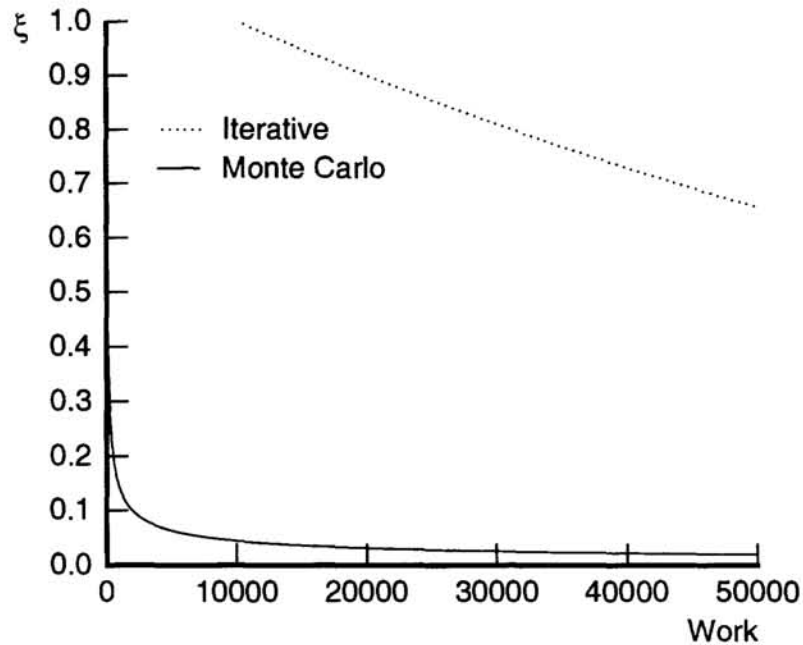

Figure 3: Error reduction versus work for $\gamma = .9$ and $n = 100$.

tion method for solving (1).

On-line RL methods for solving Markov Decision Problems, such as Real-Time Dynamic Programming (RTDP)(Barto *et. al.*, 1991), share key features with the Monte Carlo method. As with many algorithms, RTDP does not require explicit knowledge of the transition matrix, $P$, and neither, of course, do the Monte Carlo algorithms. RTDP approximates solutions to Markov Decision Problems through repeated trials of simulated or actual control, focusing computation upon regions of the state space likely to be relevant in actual control. This computational "focusing" is also a feature of the Monte Carlo algorithms. While it is true that a focusing of sorts is exhibited by Monte Carlo algorithms in an obvious way by virtue of the fact that they can compute approximate solutions for *single* components of solution vectors without exerting the computational labor required to compute all solution components, a more subtle form of computational focusing also occurs. Some of the terms in the Neumann series (2) may be very unimportant and need not be represented in the statistical estimator at all. The Monte Carlo method's stochastic estimation process achieves this automatically by, in effect, making the appearance of the representative of a non-essential term a very rare event.

These correspondences—between TD-0 and stochastic approximation, between TD-$\lambda$ and Monte Carlo methods with variance reduction, between DP-based RL algorithms for solving Markov Decision Problems and Monte Carlo algorithms — together with the comparatively favorable scaling and convergence properties enjoyed by the simple Monte Carlo method discussed in this paper, suggest that DP-based RL methods like TD/stochastic-approximation or RTDP, though perceived to be slow, may actually be advantageous for problems having a sufficiently large

number of states.

## Acknowledgement

This material is based upon work supported by the National Science Foundation under Grant ECS-9214866.

## Footnotes

[1]That is, for the iterative method, $\xi$ is defined via $\|V^{(\infty)} - V^{(n)}\| < \xi\|V^{(\infty)} - V^{(0)}\|$, while for the Monte Carlo method, $\xi$ is defined via $|V^{(\infty)}(i) - \bar{V}_M| < \xi\|V^{(\infty)} - V^{(0)}\|$, where $\bar{V}_M$ is the average over $M$ sample $V$'s.

## References

E. Barnard. Temporal-Difference Methods and Markov Models. *Submitted for publication.*

A. Barto, S. Bradtke, & S. Singh. (1991) Real-Time Learning and Control Using Asynchronous Dynamic Programming. Computer Science Department, University of Massachusetts, Tech. Rept. 91-57.

J. Curtiss. (1954) A Theoretical Comparison of the Efficiencies of Two Classical Methods and a Monte Carlo Method for Computing One Component of the Solution of a Set of Linear Algebraic Equations. In H. A. Mayer (ed.), *Symposium on Monte Carlo Methods*, 191-233. New york, NY: Wiley.

M. Duff. A Control Variate Perspective for the Optimal Weighting of Truncated, Corrected Returns. *In Preparation.*

S. Forsythe & R. Leibler. (1950) Matrix Inversion by a Monte Carlo Method. *Math. Tables Other Aids Comput.*, 4:127-129.

R. Howard. (1960) *Dynamic Programming and Markov Proceses.* Cambridge, MA: MIT Press.

R. Rubinstein. (1981) *Simulation and the Monte Carlo Method.* New York, NY: Wiley.

R. Sutton. (1988) Learning to Predict by the Method of Temporal Differences. *Machine Learning* 3:9-44.

G. Tesauro. (1992) Practical Issues in Temporal Difference Learning. *Machine Learning* 8:257-277.

W. Wasow. (1952) A Note on the Inversion of Matrices by Random Walks. *Math. Tables Other Aids Comput.*, 6:78-81.